# Density Level Detection is Classification

**Ingo Steinwart, Don Hush and Clint Scovel**
Modeling, Algorithms and Informatics Group, CCS-3
Los Alamos National Laboratory
{ingo,dhush,jcs}@lanl.gov

## Abstract

We show that anomaly detection can be interpreted as a binary classification problem. Using this interpretation we propose a support vector machine (SVM) for anomaly detection. We then present some theoretical results which include consistency and learning rates. Finally, we experimentally compare our SVM with the standard one-class SVM.

## 1   Introduction

One of the most common ways to define anomalies is by saying that *anomalies are not concentrated* (see e.g. [1, 2]). To make this precise let $Q$ be our *unknown* data-generating distribution on the input space $X$. Furthermore, to describe the concentration of $Q$ we need a *known* reference distribution $\mu$ on $X$. Let us assume that $Q$ has a density $h$ with respect to $\mu$. Then, the sets $\{h > \rho\}$, $\rho > 0$, describe the concentration of $Q$. Consequently, to define anomalies in terms of the concentration we only have to fix a threshold level $\rho > 0$, so that an $x \in X$ is considered to be anomalous whenever $x \in \{h \leq \rho\}$. Therefore our goal is to find the density level set $\{h \leq \rho\}$, or equivalently, the *$\rho$-level set $\{h > \rho\}$*. Note that there is also a modification of this problem where $\mu$ is not known but can be sampled from. We will see that our proposed method can handle both problems.

Finding density level sets is an old problem in statistics which also has some interesting applications (see e.g. [3, 4, 5, 6]) other than anomaly detection. Furthermore, a mathematical framework similar to classical PAC-learning has been proposed in [7]. Despite this effort, no *efficient* algorithm is known, which is *a)* consistent, i.e. it always finds the level set of interest asymptotically, and *b)* learns with fast rates under *realistic* assumptions on $h$ and $\mu$. In this work we propose such an algorithm which is based on an SVM approach.

Let us now introduce some mathematical notions. We begin with emphasizing that—as in many other papers (see e.g. [5] and [6])—we always assume $\mu(\{h = \rho\}) = 0$. Now, let $T = (x_1, \ldots, x_n) \in X^n$ be a training set which is i.i.d. according to $Q$. Then, a density level detection algorithm constructs a function $f_T : X \to \mathbb{R}$ such that the set $\{f_T > 0\}$ is an estimate of the $\rho$-level set $\{h > \rho\}$ of interest. Since in general $\{f_T > 0\}$ does not exactly coincide with $\{h > \rho\}$ we need a *performance measure* which describes how well $\{f_T > 0\}$ approximates the set $\{h > \rho\}$. Probably the best known performance measure (see e.g. [6, 7] and the references therein) for measurable functions $f : X \to \mathbb{R}$ is

$$\mathcal{S}_{\mu,h,\rho}(f) := \mu\Big(\{f > 0\} \triangle \{h > \rho\}\Big),$$

where $\triangle$ denotes the symmetric difference. Obviously, the smaller $\mathcal{S}_{\mu,h,\rho}(f)$ is, the more $\{f > 0\}$ coincides with the $\rho$-level set of $h$, and a function $f$ minimizes $\mathcal{S}_{\mu,h,\rho}$ if and only if $\{f > 0\}$ is $\mu$-almost surely identical to $\{h > \rho\}$. Furthermore, for a sequence of functions $f_n : X \to \mathbb{R}$ with $\mathcal{S}_{\mu,h,\rho}(f_n) \to 0$ we easily see that $\operatorname{sign} f_n \to \mathbf{1}_{\{h>\rho\}}$ both $\mu$-almost and $Q$-almost surely if $\mathbf{1}_A$ denotes the indicator function of a set $A$. Finally, it is important to note, that the performance measure $\mathcal{S}_{\mu,h,\rho}$ is somehow natural in that it is insensitive to $\mu$-zero sets.

## 2 Detecting density levels is a classification problem

In this section we show how the density level detection (DLD) problem can be formulated as a binary classification problem. To this end we write $Y := \{-1, 1\}$ and define:

**Definition 2.1** Let $\mu$ and $Q$ be probability measures on $X$ and $s \in (0, 1)$. Then the probability measure $Q \ominus_s \mu$ on $X \times Y$ is defined by

$$Q \ominus_s \mu\,(A) \;:=\; s\mathbb{E}_{x \sim Q}\mathbf{1}_A(x, 1) + (1 - s)\mathbb{E}_{x \sim \mu}\mathbf{1}_A(x, -1)$$

for all measurable $A \subset X \times Y$. Here we used the shorthand $\mathbf{1}_A(x, y) := \mathbf{1}_A((x, y))$.

Obviously, the measure $P := Q \ominus_s \mu$ can be associated with a binary classification problem in which positive samples are drawn from $Q$ and negative samples are drawn from $\mu$. Inspired by this interpretation let us recall that the binary classification risk for a measurable function $f : X \to \mathbb{R}$ and a distribution $P$ on $X \times Y$ is defined by

$$\mathcal{R}_P(f) \;=\; P\big(\{(x, y) : \operatorname{sign} f(x) \neq y\}\big),$$

where we define $\operatorname{sign} t := 1$ if $t > 0$ and $\operatorname{sign} t = -1$ otherwise. Furthermore, we denote the *Bayes risk* of $P$ by $\mathcal{R}_P := \inf\{\mathcal{R}_P(f) \,\big|\, f : X \to \mathbb{R} \text{ measurable}\}$. We will show that learning with respect to $\mathcal{S}_{\mu,h,\rho}$ is equivalent to learning with respect to $\mathcal{R}_P(.)$. To this end we begin with the following easy to prove but fundamental proposition:

**Proposition 2.2** *Let $\mu$ and $Q$ be probability measures on $X$ such that $Q$ has a density $h$ with respect to $\mu$, and let $s \in (0, 1)$. Then the marginal distribution of $P := Q \ominus_s \mu$ on $X$ is $P_X = sQ + (1 - s)\mu$. Furthermore, we $P_X$-a.s. have*

$$P(y = 1|x) = \frac{sh(x)}{sh(x) + 1 - s}.$$

Note that the above formula for $P_X$ implies that the $\mu$-zero sets of $X$ are exactly the $P_X$-zero sets of $X$. Furthermore, Proposition 2.2 shows that every distribution $P := Q \ominus_s \mu$ with $dQ := hd\mu$ and $s \in (0, 1)$ determines a triple $(\mu, h, \rho)$ with $\rho := (1 - s)/s$ and vice-versa. In the following we therefore use the shorthand $\mathcal{S}_P(f) := \mathcal{S}_{\mu,h,\rho}(f)$. Let us now compare $\mathcal{R}_P(.)$ with $\mathcal{S}_P(.)$. To this end we first observe that $h(x) > \rho = \frac{1-s}{s}$ is equivalent to $\frac{sh(x)}{sh(x)+1-s} > \frac{1}{2}$. By Proposition 2.2 the latter is $\mu$-almost surely equivalent to $\eta(x) := P(y = 1|x) > 1/2$ and hence $\mu(\{\eta > 1/2\} \triangle \{h > \rho\}) = 0$. Now recall, that binary classification aims to discriminate $\{\eta > 1/2\}$ from $\{\eta < 1/2\}$. Thus it is no surprise that $\mathcal{R}_P(.)$ can serve as a performance measure as the following theorem shows:

**Theorem 2.3** *Let $\mu$ and $Q$ be distributions on $X$ such that $Q$ has a density $h$ with respect to $\mu$. Let $\rho > 0$ satisfy $\mu(\{h = \rho\}) = 0$. We write $s := \frac{1}{1+\rho}$ and define $P := Q \ominus_s \mu$. Then for all sequences $(f_n)$ of measurable functions $f_n : X \to \mathbb{R}$ the following are equivalent:*

*i) $\mathcal{S}_P(f_n) \to 0$.*

ii) $\mathcal{R}_P(f_n) \to \mathcal{R}_P$.

*In particular, for measurable $f : X \to \mathbb{R}$ we have $\mathcal{S}_P(f) = 0$ if and only if $\mathcal{R}_P(f) = \mathcal{R}_P$.*

***Proof:*** For $n \in \mathbb{N}$ we define $E_n := \{f_n > 0\} \bigtriangleup \{h > \rho\}$. Since $\mu(\{h > \rho\} \bigtriangleup \{\eta > \frac{1}{2}\}) = 0$ it is easy to see that the classification risk of $f_n$ can be computed by

$$\mathcal{R}_P(f_n) \;=\; \mathcal{R}_P + \int_{E_n} |2\eta - 1| dP_X \,. \tag{1}$$

Now, $\{|2\eta - 1| = 0\}$ is a $\mu$-zero set and hence a $P_X$-zero set. This implies that the measures $|2\eta - 1| dP_X$ and $P_X$ are absolutely continuous with respect to each other. Furthermore, we have already observed after Proposition 2.2 that $P_X$ and $\mu$ are absolutely continuous with respect to each other. Now, the assertion follows from $\mathcal{S}_P(f_n) = \mu(E_n)$. ∎

Theorem 2.3 shows that instead of using $\mathcal{S}_P(.)$ as a performance measure for the DLD problem one can alternatively use the classification risk $\mathcal{R}_P(.)$. Therefore, we will establish some basic properties of this performance measure in the following. To this end we write $I(y, t) := \mathbf{1}_{(-\infty, 0]}(yt)$, $y \in Y$ and $t \in \mathbb{R}$, for the standard classification loss function. With this notation we can easily compute $\mathcal{R}_P(f)$:

**Proposition 2.4** *Let $\mu$ and $Q$ be probability measures on $X$. For $\rho > 0$ we write $s := \frac{1}{1+\rho}$ and define $P := Q \ominus_s \mu$. Then for all measurable $f : X \to \mathbb{R}$ we have*

$$\mathcal{R}_P(f) \;=\; \frac{1}{1+\rho} \mathbb{E}_Q I(1, \operatorname{sign} f) + \frac{\rho}{1+\rho} \mathbb{E}_\mu I(-1, \operatorname{sign} f) \,.$$

It is interesting that the classification risk $\mathcal{R}_P(.)$ is strongly connected with another approach for the DLD problem which is based on the so-called *excess mass* (see e.g. [4], [5], [6], and the references therein). To be more precise let us first recall that the excess mass of a measurable function $f : X \to \mathbb{R}$ is defined by

$$\mathcal{E}_P(f) \;:=\; Q(\{f > 0\}) - \rho\mu(\{f > 0\}) \,,$$

where $Q$, $\rho$ and $\mu$ have the usual meaning. The following proposition, that shows that $\mathcal{R}_P(.)$ and $\mathcal{E}_P(.)$ are essentially the same, can be easily checked:

**Proposition 2.5** *Let $\mu$ and $Q$ be probability measures on $X$. For $\rho > 0$ we write $s := \frac{1}{1+\rho}$ and define $P := Q \ominus_s \mu$. Then for all measurable $f : X \to \mathbb{R}$ we have*

$$\mathcal{E}_P(f) \;=\; 1 - (1 + \rho)\mathcal{R}_P(f) \,.$$

If $Q$ is an empirical measure based on a training set $T$ in the definition of $\mathcal{E}_P(.)$ we obtain the *empirical excess mass* which we denote by $\mathcal{E}_T(.)$. Then given a function class $\mathcal{F}$ the (empirical) excess mass approach chooses a function $f_T \in \mathcal{F}$ which maximizes $\mathcal{E}_T(.)$ within $\mathcal{F}$. Since the above proposition shows

$$\mathcal{E}_T(f) = 1 - \frac{1}{n} \sum_{i=1}^n I(1, \operatorname{sign} f(x_i)) - \rho\mathbb{E}_\mu I(-1, \operatorname{sign} f) \,.$$

we see that this approach is actually a type of empirical risk minimization (ERM).

In the above mentioned papers the analysis of the excess mass approach needs an additional assumption on the behaviour of $h$ around the level $\rho$. Since this condition can be used to establish a quantified version of Theorem 2.3 we will recall it now.

**Definition 2.6** Let $\mu$ be a distribution on $X$ and $h : X \to [0, \infty)$ be a measurable function with $\int h d\mu = 1$, i.e. $h$ is a density with respect to $\mu$. For $\rho > 0$ and $0 \le q \le \infty$ we say that $h$ is of $\rho$-*exponent* $q$ if there exists a constant $C > 0$ such that for all sufficiently small $t > 0$ we have

$$\mu\big(\{|h - \rho| \le t\}\big) \le Ct^q. \tag{2}$$

Condition (2) was first considered in [5, Thm. 3.6.]. This paper also provides an example of a class of densities on $\mathbb{R}^d$, $d \ge 2$, which has exponent $q = 1$. Later, Tsybakov [6, p. 956] used (2) for the analysis of a DLD method which is based on a localized version of the empirical excess mass approach. Surprisingly, (2) is satisfied if and only if $P := Q \ominus_s \mu$ has Tsybakov exponent $q$ in the sense of [8], i.e.

$$P_X\big(|2\eta - 1| \le t\big) \le C \cdot t^q \tag{3}$$

for some constant $C > 0$ and all sufficiently small $t > 0$ (see the proof of Theorem 2.7 for (2) $\Rightarrow$ (3) and [9] for the other direction). Recall that recently (3) has played a crucial role for establishing learning rates faster than $n^{-\frac{1}{2}}$ for ERM algorithms and SVM's (see e.g. [10] and [8]). Remarkably, it was already observed in [11], that the classification problem can be analyzed by methods originally developed for the DLD problem. However, to our best knowledge the exact relation between the DLD problem and binary classification has not been presented, yet. In particular, it has not been observed yet, that this relation opens a *non-heuristic* way to use classification methods for the DLD problem as we will demonstrate by example in the next section.

Let us now use the $\rho$-exponent to establish *inequalities* between $\mathcal{S}_P(.)$ and $\mathcal{R}_P(.)$:

**Theorem 2.7** *Let $\rho > 0$ and $\mu$ and $Q$ be probability measures on $X$ such that $Q$ has a density $h$ with respect to $\mu$. For $s := \frac{1}{1+\rho}$ we write $P := Q \ominus_s \mu$. Then we have*

*i) If $h$ is bounded there is a $c > 0$ such that for all measurable $f : X \to \mathbb{R}$ we have*

$$\mathcal{R}_P(f) - \mathcal{R}_P \le c\mathcal{S}_P(f).$$

*ii) If $h$ has $\rho$-exponent $q$ there is a $c > 0$ such that for all measurable $f : X \to \mathbb{R}$ we have*

$$\mathcal{S}_P(f) \le c\big(\mathcal{R}_P(f) - \mathcal{R}_P\big)^{\frac{q}{1+q}}.$$

***Sketch of the proof:*** The first assertion directly follows from (1) and Proposition 2.2. For the second assertion we first show (2) $\Rightarrow$ (3). To this end we observe that for $0 < t < \frac{1}{2}$ we have $Q\big(\{|h - \rho| \le t\}\big) \le (1 + \rho)\mu\big(\{|h - \rho| \le t\}\big)$. Thus there exists a $\tilde{C} > 0$ such that $P_X\big(\{|h - \rho| \le t\}\big) \le \tilde{C}t^q$ for all $0 < t < \frac{1}{2}$. Furthermore, $|2\eta - 1| = \big|\frac{h-\rho}{h+\rho}\big|$ implies

$$\big\{|2\eta - 1| \le t\big\} = \Big\{\frac{1 - t}{1 + t}\rho \le h \le \frac{1 + t}{1 - t}\rho\Big\},$$

whenever $0 < t < \frac{1}{2}$. Let us now define $t_l := \frac{2t}{1+t}$ and $t_r := \frac{2t}{1-t}$. This gives $1 - t_l = \frac{1-t}{1+t}$ and $1 + t_r = \frac{1+t}{1-t}$. Furthermore, we obviously also have $t_l \le t_r$. Therefore we find

$$\Big\{\frac{1 - t}{1 + t}\rho \le h \le \frac{1 + t}{1 - t}\rho\Big\} \subset \big\{|h - \rho| \le t_r\rho\big\},$$

which shows (3). Now the assertion follows from [10, Prop. 1]. ∎

## 3 A support vector machine for density level detection

One of the benefits of interpreting the DLD problem as a classification problem is that we can construct an SVM for the DLD problem. To this end let $k : X \times X \to \mathbb{R}$ be a positive

definite kernel with reproducing kernel Hilbert space (RKHS) $H$. Furthermore, let $\mu$ be a known probability measure on $X$ and $l : Y \times \mathbb{R} \to [0, \infty)$ be the *hinge* loss function, i.e. $l(y, t) := \max\{0, 1 - yt\}$, $y \in Y$, $t \in \mathbb{R}$. Then for a training set $T = (x_1, \ldots, x_n) \in X^n$, a regularization parameter $\lambda > 0$, and $\rho > 0$ our SVM for the DLD problem chooses a pair $(f_{T,\mu,\lambda}, b_{T,\mu,\lambda}) \in H \times \mathbb{R}$ which minimizes

$$\lambda \|f\|_H^2 + \frac{1}{(1 + \rho)n} \sum_{i=1}^{n} l(1, f(x_i) + b) + \frac{\rho}{1 + \rho} \mathbb{E}_{x \sim \mu} l(-1, f(x) + b) \qquad (4)$$

in $H \times \mathbb{R}$. The corresponding decision function of this SVM is $f_{T,\mu,\lambda} + b_{T,\mu,\lambda} : X \to \mathbb{R}$.

Although the measure $\mu$ is known, almost always the expectation $\mathbb{E}_{x \sim \mu} l(-1, f(x))$ can be only numerically approximated by using finitely many function evaluations of $f$. Unfortunately, since the hinge loss is not differentiable we do not know a deterministic method to choose these function evaluations efficiently. Therefore in the following we will use points $T' := (z_1, \ldots, z_m)$ which are randomly sampled from $\mu$ in order to approximate $\mathbb{E}_{x \sim \mu} l(-1, f(x))$. We denote the corresponding approximate solution of (4) by $(f_{T,T',\lambda}, b_{T,T',\lambda})$. Since this modification of (4) is identical to the standard SVM formulation besides the weighting factors in front of the empirical $l$-risk terms we do not discuss algorithmic issues. However note that this approach simultaneously addresses the original *"$\mu$ is known"* and the modified *"$\mu$ can be sampled from"* problems described in the introduction. Furthermore it is also closely related to some heuristic methods for anomaly detection that are based on artificial samples (see [9] for more information).

The fact that the SVM for DLD essentially coincides with the standard L1-SVM also allows us to modify many known results for these algorithms. For simplicity we will only state results for the Gaussian RBF kernel with width $1/\sigma$, i.e. $k(x, x') = \exp(-\sigma^2 \|x - x'\|_2^2)$, $x, x' \in \mathbb{R}^d$, and the case $m = n$. More general results can be found in [12, 9]. We begin with a consistency result with respect to the performance measure $\mathcal{R}_P(.)$. Recall that by Theorem 2.3 this is equivalent to consistency with respect to $\mathcal{S}_P(.)$:

**Theorem 3.1** *Let $X \subset \mathbb{R}^d$ be compact and $k$ be the Gaussian RBF kernel with width $1/\sigma$ on $X$. Furthermore, let $\rho > 0$, and $\mu$ and $Q$ be distributions on $X$ such that $Q$ has a density $h$ with respect to $\mu$. For $s := \frac{1}{1+\rho}$ we write $P := Q \ominus_s \mu$. Then for all positive sequences $(\lambda_n)$ with $\lambda_n \to 0$ and $n\lambda_n^{1+\delta} \to \infty$ for some $\delta > 0$, and for all $\varepsilon > 0$ we have*

$$\lim_{n \to \infty} (Q \otimes \mu)^n \Big( (T, T') \in (X \times X)^n : \mathcal{R}_P(f_{T,T',\lambda} + b_{T,T',\lambda}) > \mathcal{R}_P + \varepsilon \Big) = 0.$$

***Sketch of the proof:*** Let us introduce the shorthand $\nu = Q \otimes \mu$ for the product measure of $Q$ and $\mu$. Moreover, for a measurable function $f : X \to \mathbb{R}$ we define the function $g \circ f : X \times X \to \mathbb{R}$ by

$$g \circ f(x, x') := \frac{1}{1 + \rho} l(1, f(x)) + \frac{\rho}{1 + \rho} l(-1, f(x')), \qquad x, x' \in X.$$

Furthermore, we write $l \circ f(x, y) := l(y, f(x))$, $x \in X$, $y \in Y$. Then it is easy to check that we always have $\mathbb{E}_\nu g \circ f = \mathbb{E}_P l \circ f$. Analogously, we see $\mathbb{E}_{T \otimes T'} g \circ f = \mathbb{E}_{T \ominus_s T'} l \circ f$ if $T \otimes T'$ denotes the product measure of the empirical measures based on $T$ and $T'$. Now, using Hoeffding's inequality for $\nu$ it is easy to establish a concentration inequality in the sense of [13, Lem. III.5]. The rest of the proof is analogous to the steps in [13] since these steps are independent of the specific structure of the data-generating measure. ∎

In general, we cannot obtain convergence rates in the above theorem without assuming specific conditions on $h$, $\rho$, and $\mu$. We will now present such a condition which can be used to establish rates. To this end we write

$$\tau_x := \begin{cases} d(x, \{h > \rho\}) & \text{if } x \in \{h < \rho\} \\ d(x, \{h < \rho\}) & \text{if } x \in \{h \geq \rho\}, \end{cases}$$

where $d(x, A)$ denotes the Euclidian distance between $x$ and a set $A$. Now we define:

**Definition 3.2** Let $\mu$ be a distribution on $X \subset \mathbb{R}^d$ and $h : X \to [0, \infty)$ be a measurable function with $\int h \, d\mu = 1$, i.e. $h$ is a density with respect to $\mu$. For $\rho > 0$ and $0 < \alpha \leq \infty$ we say that $h$ has *geometric $\rho$-exponent $\alpha$* if

$$\int_X \tau_x^{-\alpha d} |h - \rho| \, d\mu \;<\; \infty \,.$$

Since $\{h > \rho\}$ and $\{h \leq \rho\}$ are the classes which have to be discriminated when interpreting the DLD problem as a classification problem it is easy to check by Proposition 2.2 that $h$ has geometric $\rho$-exponent $\alpha$ if and only if for $P := Q \ominus_s \mu$ we have $(x \mapsto \tau_x^{-1}) \in L_{\alpha d}(|2\eta - 1|dP_X)$. The latter is a sufficient condition for $P$ to have geometric noise exponent $\alpha$ in the sense of [8]. We can now state our result on learning rates which is proved in [12].

**Theorem 3.3** *Let $X$ be the closed unit ball of the Euclidian space $\mathbb{R}^d$, and $\mu$ and $Q$ be distributions on $X$ such that $dQ = h \, d\mu$. For fixed $\rho > 0$ assume that the density $h$ has both $\rho$-exponent $0 < q \leq \infty$ and geometric $\rho$-exponent $0 < \alpha < \infty$. We define*

$$\lambda_n \;:=\; \begin{cases} n^{-\frac{\alpha+1}{2\alpha+1}} & \text{if } \alpha \leq \frac{q+2}{2q} \\ n^{-\frac{2(\alpha+1)(q+1)}{2\alpha(q+2)+3q+4}} & \text{otherwise}\,, \end{cases}$$

*and $\sigma_n := \lambda_n^{-\frac{1}{(\alpha+1)d}}$ in both cases. For $s := \frac{1}{1+\rho}$ we write $P := Q \ominus_s \mu$. Then for all $\varepsilon > 0$ there exists a constant $C > 0$ such that for all $x \geq 1$ and all $n \geq 1$ the SVM using $\lambda_n$ and Gaussian RBF kernel with width $1/\sigma_n$ satisfies*

$$(Q \otimes \mu)^n \Big( (T, T') \in (X \times X)^n : \mathcal{R}_P(f_{T,T',\lambda} + b_{T,T',\lambda}) > \mathcal{R}_P + Cx^2 n^{-\frac{\alpha}{2\alpha+1}+\varepsilon} \Big) \leq e^{-x}$$

*if $\alpha \leq \frac{q+2}{2q}$ and*

$$(Q \otimes \mu)^n \Big( (T, T') \in X^{2n} : \mathcal{R}_P(f_{T,T',\lambda} + b_{T,T',\lambda}) > \mathcal{R}_P + Cx^2 n^{-\frac{2\alpha(q+1)}{2\alpha(q+2)+3q+4}+\varepsilon} \Big) \leq e^{-x}$$

*otherwise. If $\alpha = \infty$ the latter holds if $\sigma_n = \sigma$ is a constant with $\sigma > 2\sqrt{d}$.*

**Remark 3.4** With the help of Theorem 2.7 we immediately obtain rates with respect to the performance measure $\mathcal{S}_P(.)$. It turns out that these rates are very similar to those in [5] and [6] for the empirical excess mass approach.

## 4 Experiments

We present experimental results for anomaly detection problems where the set $X$ is a subset of $\mathbb{R}^d$. Two SVM type learning algorithms are used to produce functions $f$ which declare the set $\{x : f(x) < 0\}$ anomalous. These algorithms are compared based on their risk $\mathcal{R}_P(f)$. The data in each problem is partitioned into three pairs of sets; the training sets $(T, T')$, the validation sets $(V, V')$ and the test sets $(W, W')$. The sets $T$, $V$ and $W$ contain samples drawn from $Q$ and the sets $T'$, $V'$ and $W'$ contain samples drawn from $\mu$. The training and validation sets are used to design $f$ and the test sets are used to estimate its performance by computing an empirical version of $\mathcal{R}_P(f)$ that we denote $\mathcal{R}_{(W,W')}(f)$.

The first learning algorithm is the density level detection support vector machine (DLD–SVM) with Gaussian RBF kernel described in the previous section. With $\lambda$ and $\sigma^2$ fixed

and the expected value $\mathbb{E}_{x \sim \mu} l(-1, f(x) + b)$ in (4) replaced with an empirical estimate based on $T'$ this formulation can be solved using, for example, the C–SVC option in the `LIBSVM` software [14] by setting $C = 1$ and setting the class weights to $w_1 = 1/\big(|T|(1 + \rho)\big)$ and $w_{-1} = \rho/\big(|T'|(1 + \rho)\big)$. The regularization parameters $\lambda$ and $\sigma^2$ are chosen to (approximately) minimize the empirical risk $\mathcal{R}_{(V,V')}(f)$ on the validation sets. This is accomplished by employing a grid search over $\lambda$ and a combined grid/iterative search over $\sigma^2$. In particular, for each fixed grid value of $\lambda$ we seek a minimizer over $\sigma^2$ by evaluating the validation risk at a coarse grid of $\sigma^2$ values and then performing a Golden search over the interval defined by the two $\sigma^2$ values on either side of the coarse grid minimum. As the overall search proceeds the $(\lambda, \sigma^2)$ pair with the smallest validation risk is retained.

The second learning algorithm is the one–class support vector machine (1CLASS–SVM) introduced by Schölkopf et al. [15]. Due to its "one–class" nature this method does not use the set $T'$ in the production of $f$. Again we employ the Gaussian RBF kernel with parameter $\sigma^2$. The one–class algorithm in Schölkopf et al. contains a parameter $\nu$ which controls the size of the set $\{x \in T : f(x) \leq 0\}$ (and therefore controls the measure $Q(f \leq 0)$ through generalization). To make a valid comparison with the DLD–SVM we determine $\nu$ automatically as a function of $\rho$. In particular both $\nu$ and $\sigma^2$ are chosen to (approximately) minimize the validation risk using the search procedure described above for the DLD–SVM where the grid search for $\lambda$ is replaced by a Golden search (over $[0, 1]$) for $\nu$.

Data for the first experiment are generated as follows. Samples of the random variable $x \sim Q$ are generated by transforming samples of the random variable $u$ that is uniformly distributed over $[0, 1]^{27}$. The transform is $x = Au$ where $A$ is a 10–by–27 matrix whose rows contain between $m = 2$ and $m = 5$ non-zero entries with value $1/m$. Thus the support of $Q$ is the hypercube $[0, 1]^{10}$ and $Q$ is concentrated about its centers. Partial overlap in the nonzero entries across the rows of $A$ guarantee that the components of $x$ are partially correlated. We chose $\mu$ to be the uniform distribution over $[0, 1]^{10}$. Data for the second experiment are identical to the first except that the vector $(0, 0, 0, 0, 0, 0, 0, 0, 0, 1)$ is added to the samples of $x$ with probability 0.5. This gives a bi-modal distribution $Q$ and since the support of the last component is extended to $[0, 2]$ the corresponding component of $\mu$ is also extended to this range. The training and validation set sizes are $|T| = 1000$, $|T'| = 2000$, $|V| = 500$, and $|V'| = 2000$. The test set sizes $|W| = 100,000$ and $|W'| = 100,000$ are large enough to provide very accurate estimates of risk. The $\lambda$ grid for the DLD–SVM method consists of 15 values ranging from $10^{-7}$ to 1 and the coarse $\sigma^2$ grid for the DLD–SVM and 1CLASS–SVM methods consists of 9 values that range from $10^{-3}$ to $10^2$. The learning algorithms are applied for values of $\rho$ ranging from $10^{-2}$ to $10^2$. Figure 1(a) plots the risk $R_{(W,W')}$ versus $\rho$ for the two learning algorithms. In both experiments the performance of DLD–SVM is superior to 1CLASS–SVM at smaller values of $\rho$. The difference in the bi–modal case is substantial. Comparisons for larger sizes of $|T|$ and $|V|$ yield similar results, but at smaller sample sizes the superiority of DLD–SVM is even more pronounced. This is significant because $\rho \gg 1$ appears to have little utility in the general anomaly detection problem since it defines anomalies in regions where the concentration of $Q$ is much larger than the concentration of $\mu$, which is contrary to our premise that anomalies are not concentrated.

The third experiment considers a real world application in cybersecurity. The goal is to monitor the network traffic of a computer and determine when it exhibits anomalous behavior. The data for this experiment was collected from an active computer in a normal working environment over the course of 16 months. Twelve features were computed over each 1 hour time frame to give a total of 11664 12–dimensional feature vectors. These features are normalized to the range $[0, 1]$ and treated as samples from $Q$. We chose $\mu$ to be the uniform distribution over $[0, 1]^{12}$. The training, validation and test set sizes are $|T| = 4000$, $|T'| = 10000$, $|V| = 2000$, $|V'| = 100,000$, $|W| = 5664$ and $|W'| = 100,000$. The $\lambda$

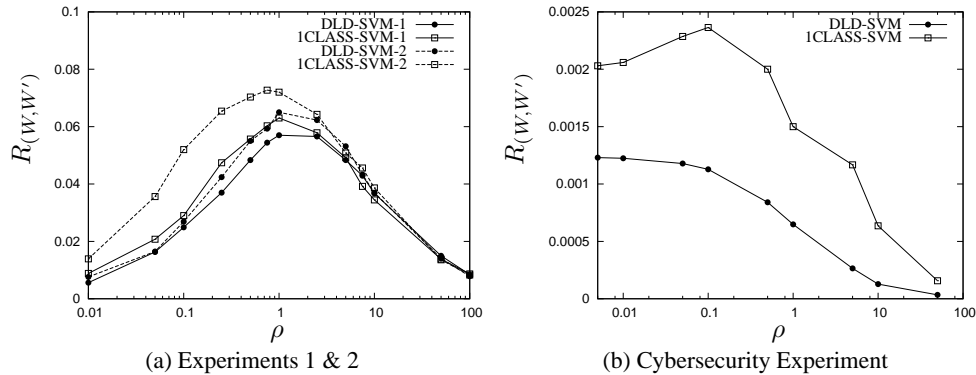

(a) Experiments 1 & 2                (b) Cybersecurity Experiment

Figure 1: Comparison of DLD–SVM and 1CLASS–SVM. The curves with extension -1 and -2 in Figure 1(a) correspond to experiments 1 and 2 respectively.

grid for the DLD–SVM method consists of 7 values ranging from $10^{-7}$ to $10^{-1}$ and the coarse $\sigma^2$ grid for the DLD–SVM and 1CLASS–SVM methods consists of 6 values that range from $10^{-3}$ to $10^2$. The learning algorithms are applied for values of $\rho$ ranging from 0.05 to 50.0. Figure 1(b) plots the risk $R_{(W,W')}$ versus $\rho$ for the two learning algorithms. The performance of DLD–SVM is superior to 1CLASS–SVM at all values of $\rho$.

## References

[1] B.D. Ripley. *Pattern recognition and neural networks*. Cambridge Univ. Press, 1996.

[2] B. Schölkopf and A.J. Smola. *Learning with Kernels*. MIT Press, 2002.

[3] J.A. Hartigan. *Clustering Algorithms*. Wiley, New York, 1975.

[4] J.A. Hartigan. Estimation of a convex density contour in 2 dimensions. *J. Amer. Statist. Assoc.*, 82:267–270, 1987.

[5] W. Polonik. Measuring mass concentrations and estimating density contour clusters—an excess mass aproach. *Ann. Stat.*, 23:855–881, 1995.

[6] A.B. Tsybakov. On nonparametric estimation of density level sets. *Ann. Statist.*, 25:948–969, 1997.

[7] S. Ben-David and M. Lindenbaum. Learning distributions by their density levels: a paradigm for learning without a teacher. *J. Comput. System Sci.*, 55:171–182, 1997.

[8] C. Scovel and I. Steinwart. Fast rates for support vector machines. *Ann. Statist.*, submitted, 2003. http://www.c3.lanl.gov/~ingo/publications/ann-03.ps.

[9] I. Steinwart, D. Hush, and C. Scovel. A classification framework for anomaly detection. Technical report, Los Alamos National Laboratory, 2004.

[10] A.B. Tsybakov. Optimal aggregation of classifiers in statistical learning. *Ann. Statist.*, 32:135–166, 2004.

[11] E. Mammen and A. Tsybakov. Smooth discrimination analysis. *Ann. Statist.*, 27:1808–1829, 1999.

[12] C. Scovel, D. Hush, and I. Steinwart. Learning rates for support vector machines for density level detection. Technical report, Los Alamos National Laboratory, 2004.

[13] I. Steinwart. Consistency of support vector machines and other regularized kernel machines. *IEEE Trans. Inform. Theory*, to appear, 2005.

[14] Chih-Chung Chang and Chih-Jen Lin. LIBSVM: a library for support vector machines, 2004.

[15] B. Schölkopf, J.C. Platt, J. Shawe-Taylor, and A.J. Smola. Estimating the support of a high-dimensional distribution. *Neural Computation*, 13:1443–1471, 2001.
